# Convergence of a Neural Network Classifier

**John S. Baras**
Systems Research Center
University of Maryland
College Park, Maryland 20705

**Anthony LaVigna**
Systems Research Center
University of Maryland
College Park, Maryland 20705

## Abstract

In this paper, we prove that the vectors in the LVQ learning algorithm converge. We do this by showing that the learning algorithm performs stochastic approximation. Convergence is then obtained by identifying the appropriate conditions on the learning rate and on the underlying statistics of the classification problem. We also present a modification to the learning algorithm which we argue results in convergence of the LVQ error to the Bayesian optimal error as the appropriate parameters become large.

## 1 Introduction

Learning Vector Quantization (LVQ) originated in the neural network community and was introduced by Kohonen (Kohonen [1986]). There have been extensive simulation studies reported in the literature demonstrating the effectiveness of LVQ as a classifier and it has generated considerable interest as the training times associated with LVQ are significantly less than those associated with backpropagation networks.

In this paper we analyse the convergence properties of LVQ. Using a theorem from the stochastic approximation literature, we prove that the update algorithm converges under the suitable conditions. We also present a modification to the algorithm which provides for more stable learning. Finally, we discuss the decision error associated with this "modified" LVQ algorithm.

## 2   A Review of Learning Vector Quantization

Let $\{(x_i, d_{x_i})\}_{i=1}^N$ be the training data or past observation set. This means that $x_i$ is observed when pattern $d_{x_i}$ is in effect. We assume that the $x_i$'s are statistically independent (this assumption can be relaxed). Let $\theta_j$ be a Voronoi vector and let $\Theta = \{\theta_1, \ldots, \theta_k\}$ be the set of Voronoi vectors. We assume that there are many more observations than Voronoi vectors (Duda & Hart [1973]). Once the Voronoi vectors are initialized, training proceeds by taking a sample $(x_j, d_{x_j})$ from the training set, finding the closest Voronoi vector and adjusting its value according to equations (1) and (2). After several passes through the data, the Voronoi vectors converge and training is complete.

Suppose $\theta_c$ is the closest vector. Adjust $\theta_c$ as follows:

$$\theta_c(n+1) = \theta_c(n) + \alpha_n \ (x_j - \theta_c(n)) \tag{1}$$

if $d_{\theta_c} = d_{x_j}$ and

$$\theta_c(n+1) = \theta_c(n) - \alpha_n \ (x_j - \theta_c(n)) \tag{2}$$

if $d_{\theta_c} \neq d_{x_j}$. The other Voronoi vectors are not modified.

This update has the effect that if $x_j$ and $\theta_c$ have the same decision then $\theta_c$ is moved closer to $x_j$, however if they have different decisions then $\theta_c$ is moved away from $x_j$. The constants $\{\alpha_n\}$ are positive and decreasing, e.g., $\alpha_n = 1/n$. We are concerned with the convergence properties of $\Theta(n)$ and with the resulting detection error.

For ease of notation, we assume that there are only two pattern classes. The equations for the case of more than two pattern classes are given in (LaVigna [1989]).

## 3   Convergence of the Learning Algorithm

The LVQ algorithm has the general form

$$\theta_i(n+1) = \theta_i(n) + \alpha_n \ \gamma(d_{x_n}, d_{\theta_i(n)}, x_n, \Theta_n) \ (x_n - \theta_i(n)) \tag{3}$$

where $x_n$ is the currently chosen past observation. The function $\gamma$ determines whether there is an update and what its sign should be and is given by

$$\gamma(d_{x_n}, d_{\theta_i}, x_n, \Theta_n) = \begin{cases} -1 & \text{if } d_{x_n} = d_{\theta_i} \text{ and } x_n \in V_{\theta_i} \\ 1 & \text{if } d_{x_n} \neq d_{\theta_i} \text{ and } x_n \in V_{\theta_i} \\ 0 & \text{otherwise} \end{cases} \tag{4}$$

Here $V_{\theta_i}$ represents the set of points closest to $\theta_i$ and is given by

$$V_{\theta_i} = \{ x \in \Re^d \ : \ \|\theta_i - x\| < \|\theta_j - x\|, \ j \neq i \} \qquad i = 1, \ldots, k. \tag{5}$$

The update in (3) is a stochastic approximation algorithm (Benveniste, Metivier & Priouret [1987]). It has the form

$$\Theta_{n+1} = \Theta_n + \alpha_n \ H(\Theta_n, z_n) \tag{6}$$

where $\Theta$ is the vector with components $\theta_i$; $H(\Theta, z)$ is the vector with components defined in the obvious manner from (3) and $z_n = (x_n, d_{x_n})$ is the random pair

consisting of the observation and the associated *true* pattern number. If the appropriate conditions are satisfied by $\alpha_n$, $H$, and $z_n$, then $\Theta_n$ approaches the solution of

$$\frac{d}{dt}\bar{\Theta}(t) = h(\bar{\Theta}(t)) \tag{7}$$

for the appropriate choice of $h(\Theta)$.

For the two pattern case, we let $p_1(x)$ represent the density for pattern 1 and $\pi_1$ represent its prior. Likewise for $p_0(x)$ and $\pi_0$. It can be shown (Kohonen [1986]) that

$$h_i(\Theta) = \int_{V_{\theta_i}} (x - \Theta_i)\, q_i(x)\, dx \tag{8}$$

where

$$q_i(\Theta) = \begin{cases} p_1(x)\pi_1 - p_0(x)\pi_0 & \text{if } d_{\theta_i} = 1 \\ p_0(x)\pi_0 - p_1(x)\pi_1 & \text{if } d_{\theta_i} = 1 \end{cases} \tag{9}$$

If the following hypotheses hold then using techniques from (Benveniste, Metivier & Priouret [1987]) or (Kushner & Clark [1978]) we can prove the convergence theorem below:

[H.1] $\{\alpha_n\}$ is a nonincreasing sequence of positive reals such that $\sum_n \alpha_n = \infty$, $\sum_n \alpha_n^\lambda < \infty$.

[H.2] Given $d_{x_n}$, $x_n$ are independent and distributed according to $p_{d_{x_n}}(x)$.

[H.3] The pattern densities, $p_i(x)$, are continuous.

**Theorem 1** *Assume that [H.1]-[H.3] hold. Let $\bar{\Theta}^*$ be a locally asymptotic stable equilibrium point of (7) with domain of attraction $D^*$. Let $Q$ be a compact subset of $D^*$. If $\Theta_n \in Q$ for infinitely many $n$ then*

$$\lim_{n \to \infty} \Theta_n = \bar{\Theta}^* \quad a.s. \tag{10}$$

**Proof:** (see (LaVigna [1989]))

Hence if the initial locations and decisions of the Voronoi vectors are close to a locally asymptotic stable equilibrium of (7) and if they do not move too much then the vectors converge.

Given the form of (8) one might try to use Lyapunov theory to prove convergence with

$$L(\Theta) = \sum_{i=1}^{K} \int_{V_{\theta_i}} \|x - \Theta_i\|^2 \, q_i(x),\, dx \tag{11}$$

as a candidate Lyapunov function. This function will not work as is demonstrated by the following calculation in the one dimensional case. Suppose that $K = 2$ and $\theta_1 < \theta_2$ then

$$\frac{\partial}{\partial \theta_1} L(\Theta) = \frac{\partial}{\partial \theta_1} \sum_{i=1}^{2} \int_{V_{\theta_i}} \|x - \Theta_i\|^2 \, q_i(x),\, dx \tag{12}$$

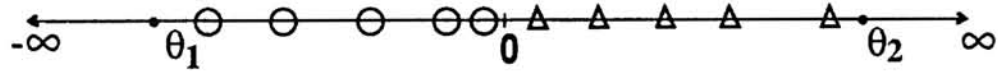

Figure 1: *A possible distribution of observations and two Voronoi vectors.*

$$= \frac{\partial}{\partial \theta_1} \left( \int_{V_{\theta_1}} \|x - \Theta_1\|^2 \, q_1(x), dx + \int_{V_{\theta_2}} \|x - \Theta_2\|^2 \, q_2(x), dx \right. \tag{13}$$

$$= \frac{\partial}{\partial \theta_1} \left( \int_{-\infty}^{(\theta_1+\theta_2)/2} \|x - \Theta_1\|^2 \, q_1(x), dx + \int_{(\theta_1+\theta_2)/2}^{\infty} \|x - \Theta_2\|^2 \, q_2(x), dx \right) \tag{14}$$

$$= -2 \int_{-\infty}^{(\theta_1+\theta_2)/2} (x - \Theta_1) \, q_1(x), dx + \|(\theta_1 - \theta_2)/2\|^2 \, q_1((\theta_1 + \theta_2)/2) \tag{15}$$

$$= -h_1(\Theta) + \|(\theta_1 - \theta_2)/2\|^2 \, q_1((\theta_1 + \theta_2)/2) \tag{16}$$

$$\tag{17}$$

Likewise

$$\frac{\partial}{\partial \theta_2} L(\Theta) = -h_2(\Theta) + \|(\theta_2 - \theta_1)/2\|^2 \, q_2((\theta_1 + \theta_2)/2) \tag{18}$$

Therefore

$$\nabla L(\Theta)\dot{\Theta} = -h_1(\Theta)^2 - h_2(\Theta)^2 + \|(\theta_1 - \theta_2)/2\|^2 \, q_1((\theta_1 + \theta_2)/2)(h_1(\Theta) - h_2(\Theta)) \tag{19}$$

In order for this to be a Lyapunov function (19) would have to be strictly nonpositive which is not the case. The problem with this candidate occurs because the integrand $q_i(x)$ is not strictly positive as is the case for ordinary vector quantization and adaptive K-means.

## 4   Modified LVQ Algorithm

The convergence results above require that the initial conditions are close to the stable points of (7) in order for the algorithm to converge. In this section we present a modification to the LVQ algorithm which increases the number of stable equilibrium for equation (7) and hence increases the chances of convergence. First we present a simple example which emphasizes a defect of LVQ and suggests an appropriate modification to the algorithm.

Let $\bigcirc$ represent an observation from pattern 2 and let $\triangle$ represent an observation from pattern 1. We assume that the observations are scalar. Figure 1 shows a possible distribution of observations. Suppose there are two Voronoi vectors $\theta_1$ and $\theta_2$ with decisions 1 and 2, respectively, initialized as shown in Figure 1. At each update of the LVQ algorithm, a point is picked at random from the observation set and the closest Voronoi vector is modified. We see that during this update, it is possible for $\theta_2(n)$ to be pushed towards $\infty$ and $\theta_1(n)$ to be pushed towards $-\infty$, hence the Voronoi vectors may not converge.

Recall that during the update procedure in (3), the Voronoi cells are changed by changing the location of one Voronoi vector. After an update, the majority vote of

the observations in each new Voronoi cell may not agree with the decision previously assigned to that cell. This discrepency can cause the divergence of the algorithm. In order to prevent this from occuring the decisions associated with the Voronoi vectors should be updated to agree with the majority vote of the observations that fall within their Voronoi cells. Let

$$g_i(\Theta; N) = \begin{cases} 1 & \text{if } \frac{1}{N} \sum_{j=1}^{N} 1_{\{y_j \in V_{\theta_i}\}} 1_{\{d_{y_j}=1\}} > \frac{1}{N} \sum_{j=1}^{N} 1_{\{y_j \in V_{\theta_i}\}} 1_{\{d_{y_j}=2\}} \\ 2 & \text{otherwise.} \end{cases} \quad (20)$$

Then $g_i$ represents the decision of the majority vote of the observations falling in $V_{\theta_i}$. With this modification, the learning for $\theta_i$ becomes

$$\theta_i(n+1) = \theta_i(n) + \alpha_n \, \gamma(d_{x_n}, g_i(\Theta_n; N), x_n, \Theta_n) \, \nabla_{\theta_i(n)}(\theta_i(n) - x_n). \quad (21)$$

This equation has the same form as (3) with the function $\overline{H}(\Theta, z)$ defined from (21) replacing $H(\Theta, z)$.

This divergence happens because the decisions of the Voronoi vectors do not agree with the majority vote of the observations closest to each vector. As a result, the Voronoi vectors are pushed away from the origin. This phenomena occurs even though the observation data is bounded. The point here is that, if the decision associated with a Voronoi vector does not agree with the majority vote of the observations closest to that vector then it is possible for the vector to diverge. A simple solution to this problem is to correct the decisions of all the Voronoi vectors after every adjustment so that their decisions correspond to the majority vote. In practice this correction would only be done during the beginning iterations of the learning algorithm since that is when $\alpha_n$ is large and the Voronoi vectors are moving around significantly. With this modification it is possible to show convergence to the Bayes optimal classifier (LaVigna [1989]) as the number of Voronoi vectors become large.

## 5    Decision Error

In this section we discuss the error associated with the modified LVQ algorithm. Here two results are discussed. The first is the simple comparison between LVQ and the nearest neighbor algorithm. The second result is if the number of Voronoi vectors is allowed to go to infinity at an appropriate rate as the number of observations goes to infinity, then it is possible to construct a convergent estimator of the Bayes risk. That is, the error associated with LVQ can be made to approach the optimal error. As before, we concentrate on the binary pattern case for ease of notation.

### 5.1    Nearest Neighbor

If a Voronoi vector is assigned to each observation then the LVQ algorithm reduces to the nearest neighbor algorithm. For that algorithm, it was shown (Cover & Hart [1967]) that its Bayes minimum probability of error is less than twice that of the optimal classifier. More specifically, let $r^*$ be the Bayes optimal risk and let $r$ be

the nearest neighbor risk. It was shown that

$$r^* \leq r \leq 2r^*(1 - r^*) \leq 2r^*. \tag{22}$$

Hence in the case of no iteration, the Bayes' risk associated with LVQ is given from the nearest neighbor algorithm.

## 5.2  Other Choices for Number of Voronoi Vectors

We saw above that if the number of Voronoi vectors equals the number of observations then LVQ coincides with the nearest neighbor algorithm. Let $k_N$ represent the number of Voronoi vectors for an observation sample size of $N$. We are interested in determining the probability of error for LVQ when $k_N$ satisfies (1) $\lim k_N = \infty$ and (2) $\lim(k_N/N) = 0$. In this case, there are more observations than vectors and hence the Voronoi vectors represent averages of the observations. It is possible to show that with $k_N$ satisfying (1)-(2) the decision error associated with modified LVQ can be made to approach the Bayesian optimal decision error as $N$ becomes large  (LaVigna [1989]).

## 6  Conclusions

We have shown convergence of the Voronoi vectors in the LVQ algorithm. We have also presented the majority vote modification of the LVQ algorithm. This modification prevents divergence of the Voronoi vectors and results in convergence for a larger set of initial conditions. In addition, with this modification it is possible to show that as the appropriate parameters go to infinity the decision regions associated with the modified LVQ algorithm approach the Bayesian optimal (LaVigna [1989]).

## 7  Acknowledgements

This work was supported by the National Science Foundation through grant CDR-8803012, Texas Instruments through a TI/SRC Fellowship and the Office of Naval Research through an ONR Fellowship.

## 8  References

A. Benveniste, M. Metivier & P. Priouret [1987], *Algorithmes Adaptatifs et Approximations Stochastiques*, Mason, Paris.

T. M. Cover & P. E. Hart [1967], "Nearest Neighbor Pattern Classification," *IEEE Transactions on Information Theory* IT-13, 21–27.

R. O. Duda & P. E. Hart [1973], *Pattern Classification and Scene Analysis*, John Wiley & Sons, New York, NY.

T. Kohonen [1986], "Learning Vector Quantization for Pattern Recognition," Technical Report TKK-F-A601, Helsinki University of Technology.

H. J. Kushner & D. S. Clark [1978], *Stochastic Approximation Methods for Constrained and Unconstrained Systems* , Springer-Verlag, New York–Heidelberg–Berlin.

A. LaVigna [1989], "Nonparametric Classification using Learning Vector Quantization," Ph.D. Dissertation, Department of Electrical Engineering, University of Maryland.